# A New Multi-Source Light Detection Benchmark and Semi-Supervised Focal Light Detection

**Jae-Yong Baek**     **Yong-Sang Yoo**     **Seung-Hwan Bae**[*]

Inha University, Dept. of Electrical and Computer Engineering, Vision and Learning Lab.
`{jy1213, cpp0094}@inha.edu, shbae@inha.ac.kr`

## Abstract

This paper addresses a multi-source light detection (LD) problem from vehicles, traffic signals, and streetlights under driving scenarios. Albeit it is crucial for autonomous driving and night vision, this problem has not been yet focused on as much as other object detection (OD). One of the main reasons is the absence of a public available LD benchmark dataset. Therefore, we construct a new large LD dataset consisting of different light sources via heavy annotation:YouTube Driving Light Detection dataset (YDLD). Compared to the existing LD datasets, our dataset has much more images and box annotations for multi-source lights. We also provide rigorous statistical analysis and transfer learning comparison of other well-known detection benchmark datasets to prove the generality of our YDLD.

For the recent object detectors, we achieve the extensive comparison results on YDLD. However, they tend to yield the low mAP scores due to the intrinsic challenges of LD caused by very tiny size and similar appearance. To resolve those, we design a novel lightness focal loss which penalizes miss-classified samples more and a lightness spatial attention prior by reflecting a global scene context. In addition, we develop a semi-supervised focal light detection (SS-FLD) by embedding our lightness focal loss into the semi-supervised object detection (SSOD). We prove that our methods can consistently boost mAP to the variety of types of recent detectors on YDLD. We will open both YDLD and SS-FLD code at https://github.com/YDLD-dataset/YDLD.

## 1 Introduction

Light detection (LD) in a driving environment is a fundamental technology for autonomous driving and night vision. During the past few years, there have been efforts to detect light sources [1, 2] or remove unnecessary light effects (*e.g.* sun glare) [2]. Nevertheless, its progress still lags behind the other detection studies [3–6] and the solid baseline for light detection is not yet designed. The main reason comes from the lack of public available multi-source LD datasets. In specific, the existing dataset is limited to the binary detection problem. However, each light source should be disentangled since they convey the different contexts of understanding an image scene.

Therefore, we release a novel multi-source light detection dataset: *YouTube Driving Light Detection (YDLD)*. Our dataset contains 3.5k images and 116k box annotations for three light sources: a vehicle light, a traffic signal, and a streetlight. We select these classes since they are important cues to predict the behaviors of other vehicles and understand the road environment. To build this dataset, we first collect a huge amount of images captured in different times and spaces from YouTube and then tailor appropriate frames which contain many light sources. The most images were captured on real driving environment at night or evening time. Then, we perform the costly bounding box annotation per image. In particular, we strive to annotate very tiny light sources carefully as shown in Fig. 1.

---

[*]Corresponding author.

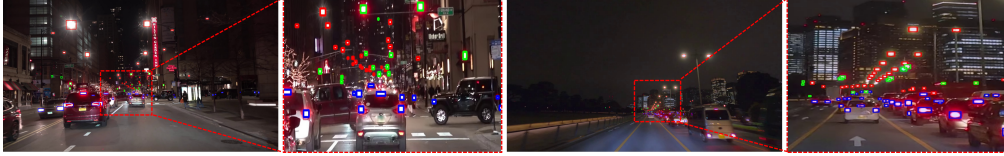

Figure 1: Examples of our YouTube Driving Light Detection (YDLD) dataset. Red, blue, and green boxes represent Streetlight, Carlight, and TrafficSignallight class, respectively. YDLD dataset consists of 3.5k night and evening driving images and 116k box annotations.

In addition, we provide the detailed statistics of our YDLD in Table 1 and 2 (Refer to our supplementary material). To show the feasibility of our YDLD as a benchmark dataset, we conduct an extensive LD comparison over many recent detectors and compare transfer learning effects of conventional OD datasets on the large BDD100k dataset. In particular, we compare different types of single-stage, two-stage, transformer and semi-supervised object detection (SSOD), tiny object-based 20 detectors as shown in Fig. 3 and Table 5.

However, the LD results for even recent detectors showing high mAP scores on the MS-COCO dataset are not satisfactory on YDLD. This is due to the intrinsic challenges in LD: very tiny size and similar appearance. More detailed comparisons and analysis are provided in the supplementary material. As a result, the extracted light feature renders non-discriminative and blurred [7–9]. In addition, frequent false positives occur due to the box mismatch over tiny lights [10].

For more accurate LD, we propose a lightness focal (LF) loss which can penalize higher false positive responses more and the lightness attention prior. The attention prior reflects the geometrical relations between a mounted camera and a light source under flat road driving. We further enhance LD by incorporating our LF loss into the SSOD. We show the effects of our method via ablation study, and consistent accuracy improvement of the conventional detectors can be achieved by our method.

The main contributions of this paper are: (i) a new benchmark dataset on multi-source light detections, (ii) a lightness focal loss penalizing false positives and embedding lightness prior, (iii) extensive evaluation over various types of recent detectors and consistent improvements by our method.

## 2    Related works

A light source detection problem is to identify and localize light sources emitted from objects. This problem is a still challenging problem. Because the class of a light source is determined from the laid object, many light sources have similar color and shape. In the real-world scenario such as driving, the light size is very tiny. So, this problem can be considered as tiny object detection (TOD) [10, 7].

**Light detection** Until now, most works on LD have been focused on the binary detection of classifying a light and other background. The early studies are based on hand-craft features [11–13]. They use pixel intensity as a main feature cue since the pixels on the light have higher intensity in general. Methods using color features [11, 12], intensity thresholding [14, 15], saturation and local contrast features [12] have been presented. For improving LD, deep learning methods [2, 1, 16] have been provided by using CNN as a feature extractor. [1] creates uncertainty maps using blob detection and CNN. [2] segments sun glare using CycleGAN [17]. [18] designs an encoder-decoder network for flare removal.

On the other hand, since flare and glare caused by light source can degrade image quality, it is important to remove them. Moreover, they highly correlate with light sources, and datasets have been proposed to learn this prior relationship. [19] creates and releases a semi-synthetic dataset by simulating the physical phenomena related to flare and synthesizing flare and glare onto clean images. Flare7K [20] focuses on glare and lens flare removal in low-light conditions, especially at night. Similar to [19], it uses synthetic flare and glare. Additionally, the more image enhancement research has been conducted in low-light conditions by utilizing light sources. [21] proposes a network that predicts the inverse camera response function to improve the visibility of low-light images while removing noise and glare. [22] proposes an unsupervised approach that combines a layer decomposition network with a light-effects suppression network. Despite these efforts for LD, understanding the characteristics of light sources remains a tricky issue, since they only focus detecting binary classes, glare, and flare.

To resolve this, we address a multi-source light detection problem. To the best of our knowledge, this work is a pioneer work in multi-source LD. This is more challenging since the contextual learning between a light and a related object is needed to recognize different light classes. Therefore, we build a new YDLD and evaluate recent detectors incorporating our methods on YDLD.

**Tiny object detection** Tiny objects refer to objects smaller than 32×32 pixels, based on the MS-COCO standard. Due to their small size, tiny object detection (TOD) is still considered as a challenging task. To address the TOD problem, various datasets and methods have been proposed. DOTA [23] (Dataset for Object Detection in Aerial Images) dataset handles the aerial image domain and includes 15 classes such as vehicles, airplanes, and roundabouts. AI-TOD (Tiny Object Detection in Aerial Images) [24] dataset also focuses on the aerial image domain and covers 8 classes, including airplanes, bridges, and swimming pools.

To address the TOD problem, tiny object-oriented methods are proposed. FSANet [25] extracts more discriminative features for tiny objects by utilizing feature-and-spatial alignment and incremental optimization techniques. RFLA [7] improves the performance of tiny object detection by proposing a Gaussian receptive-based label assignment method.

## 3 YouTube driving light detection dataset

We discuss the details of our YDLD in this section.

### 3.1 Motivation

In autonomous driving and surveillance systems, light source detection is a fundamental technology to recognize objects, predict object behaviors, and understand the road or surveillance environment. The previous works have mainly focused on binary detection that discriminates between light and another background. Indeed, even different light sources have very similar feature distributions since the pixels within the sources are usually saturated. Therefore, the other object contexts (*e.g.* poles and cars) nearby lights should be leveraged as well to identify between light sources. To achieve this, we build a new multi-source

Table 1: Overview and annotation statistics of bounding boxes of the YDLD.

| Overview of YDLD dataset | | | |
|---|---|---|---|
| | **Train** | **Test** | **Total** |
| # images | 1,588 | 1,928 | 3,516 |
| # GTs | 49,387 | 66,641 | 116,028 |
| # Carlight | 20,411 | 28,648 | 49,059 |
| # TrafficSignallight | 5,041 | 9,332 | 14,373 |
| # Streetlight | 23,935 | 28,661 | 52,596 |
| Avg. of # GTs | 31.10 | 34.56 | - |
| Highest # GTs on an image | 173 | 501 | - |
| Avg. of bbox size | $19.21 \times 14.60$ | $19.36 \times 14.64$ | - |
| Largest bbox size | $232 \times 447$ | $362 \times 650$ | - |
| **Annotation statistics of bounding boxes** | | | |
| **Bbox area** | **Range** | **Train** | **Test** | **Total** |
| Very tiny | $[0^2, 8^2)$ | 16,060 | 20,445 | 36,505 |
| Tiny | $[8^2, 16^2)$ | 17,789 | 24,610 | 42,399 |
| Small | $[16^2, 32^2)$ | 10,487 | 14,478 | 24,965 |
| Medium | $[32^2, inf)$ | 5,051 | 7,108 | 12,159 |
| Total | - | 49,387 | 66,641 | 116,028 |

LD dataset containing diverse scenarios and objects. Our dataset includes the traffic, the car, and the street lights which are often encountered during driving. We perform meticulous annotation even for very tiny objects as shown in Table 1. We also evaluate modern detectors with different architectures and discuss their pros and cons. Furthermore, we enhance the detectors by using our lightness focal loss and SSOD.

### 3.2 Data collection, annotation, and quality maintenance

For improving dataset diversity, light image collection from various environments is important but costly. To alleviate this, we collect videos from YouTube from over different 19 countries (*i.e.* United States, Italy, Japan, etc) and various driving scenarios. The videos are recorded by HD $(1,280 \times 720)$ or UHD $(3,840 \times 2,160)$ and are manually selected to exclude similar redundant images. In order to enhance the dataset generality, we capture images having different light appearances, emitting directions, and locations. For privacy protection, we blur pixels (*e.g.* , license plates, faces) containing personal information.

For each image, five people perform annotation of 2D bounding boxes and classes using the COCO annotation tool [26]. We consider three light source classes of Carlight, TrafficSignallight, and Streetlight. The carlight corresponds to a headlight, rear lamp, or break lamp of a vehicle such as a car, truck, bus, or motorcycle. The traffic signal light and streetlight are originated from a traffic signal bulb and streetlight lamp.

Table 2: Details of YDLD statistics over 19 countries.

| | Data Time | | | Roadway Type | | Traffic Light Shape | | Traffic Direction | |
|---|---|---|---|---|---|---|---|---|---|
| | Day | Evening | Night | Urban | Highway | Horizontal | Vertical | Left-handed | Right-handed |
| **Streetlight** | 263 | 6,030 | 46,300 | 45,906 | 6,687 | 19,212 | 33,381 | 10,700 | 41,893 |
| **Carlight** | 267 | 6,450 | 42,342 | 42,079 | 6,980 | 19,402 | 29,657 | 10,271 | 38,788 |
| **TrafficSignallight** | 99 | 2,350 | 11,924 | 14,285 | 88 | 4,808 | 9,565 | 2,560 | 11,813 |

Table 3: Comparison of dataset statistics with other light detection datasets.

| Dataset | # images | # annotations | # countries | Multi-source light | Light sources | Day/Night | Published |
|---|---|---|---|---|---|---|---|
| Chen et al. [2] | 39k | N/A | N/A | ✗ | Sun | Day | ✗ |
| Yahiaoui et al. [1] | 1.4k | N/A | 5 | ✗ | Sun | Day | ✗ |
| Andalibi et al. [12] | 10 | N/A | N/A | ✓ | Sun, Streetlight | Day/Night | ✗ |
| Singh et al. [13] | N/A | N/A | N/A | ✗ | Streetlight | Night | ✗ |
| Esfahani et al. [29] | 200 | 200 | N/A | ✗ | Sun | Day | ✓ |
| Yoneda et al. [30] | 9k | 1.3k | N/A | ✗ | Sun | Day | ✗ |
| YDLD | 3.5k | 116k | 19 | ✓ | Vehicle, Streetlight, Traffic light | Night | ✓ |

For accurate annotation of light sources, the following annotation policy is obeyed: (1) labeling the light source (*i.e.* bulb) itself by excluding the glare or emitted light regions (2) labeling each light source separately even if light sources are very closely located (3) excluding any reflection and interior light within a building. Annotation samples are shown in Fig. 2.

For images containing lights, inappropriate images with low-quality lights and severe artifacts (*e.g.* motion blur) are excluded by a researcher inspection phase. To improve label qualities, annotated labels are cross-checked by other researchers. From this phase, missing or inaccurate annotations are fixed. Then, the trained baseline detectors (*e.g.* PAA [27] and DINO [28]) are investigated whether their mAP scores are appropriately evaluated.

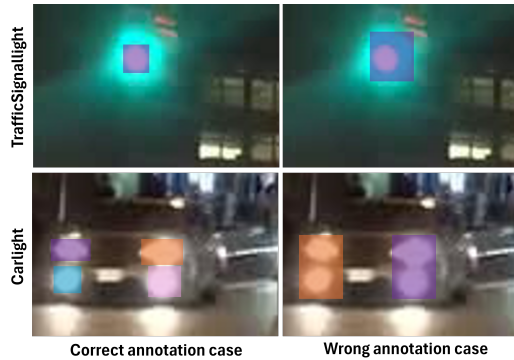

Figure 2: Labeling examples according to YDLD annotation policy.

## 3.3 Evaluation

Due to the tiny light size, we use AI-TOD detection metrics [24]. We compute average precision (AP) at IoU $\in [0.5 : 0.05 : 0.95]$ (box AP) to calculate mean AP ($mAP$), and very tiny ($AP_{vt}$), tiny ($AP_t$), small ($AP_s$), and medium ($AP_m$) scores. The object size category is defined in Table 1.

## 3.4 Data statistics

Table 1 shows the main statistics of our constructed YDLD. For the around 3,516 images, we generate 116,028 ground truth boxes. We use 49,387 and 66,641 boxes for training and testing. In Table 1, we categorize each box according to the size guideline of a tiny object detection (TOD) work [24]. Since the light source laid on an object occupies a very small area in general, the portion for tiny and very tiny object is significant to around 68%. Therefore, this LD problem is closely related to a TOD which is more challenging than the common OD. Thus, the overall mAP score is very sensitive to a localization result [10] because an IoU score is significantly affected by the variation of an intersection area between GT and predicted regions.

Table 2 presents the statistics of various attributes that affect light appearance, shape, emitting direction, and location. These statistics demonstrate that our dataset considers a diverse range of attributes and driving environments, indicating comprehensive data collection and class labeling across various driving scenarios.

The YDLD dataset primarily includes light detection (LD) during nighttime and evening hours since most lights are visible well in these time intervals. Moreover, it contains 88% of urban images showing denser light distributions and more complex vehicle interactions. The dataset contains different traffic light shapes and traffic directions which vary across countries. These various attributes improve the generality of our dataset.

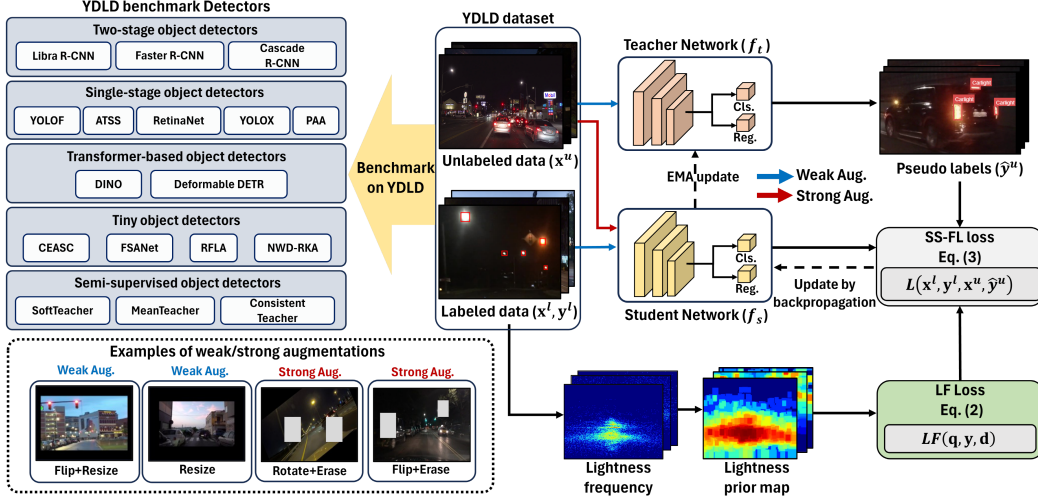

Figure 3: The YDLD benchmark evaluation: we implement and evaluate the different types of the recent 20 detectors by using our lightness focal loss and SS-FLD methods in Sec. 4.1-4.3.

## 3.5 Comparison with other light detection datasets

We compare our YDLD with other light detection datasets. Unfortunately, most datasets are not open to the public, and most contents are described based on the corresponding papers. We confirm that most datasets provide the labels of single light sources such as sun or streetlight only. In addition, the dataset scale of our YDLD is much larger than other datasets for the annotation quantity. For the aspect of image diversity, our YDLD overwhelms others since we consider many more attributes over 19 different countries as in Table 3. The most important merit of our dataset and code is publicly available, and it could expedite the development of other LD works.

## 4 Semi-supervised focal light detection

We provide the overall architecture of our light detector in Fig. 3 based on SSOD [31–33]. For improving LD, we exploit our lightness focal (LF) loss for training SSOD.

### 4.1 Lightness focal loss

The appearance similarity of light sources, the tiny size, and non-light objects exposed by high light intensity brings about a lot of false positive detections. To remedy this, we design a lightness focal loss by extending the original focal loss [34] focusing on the miss-classified hard samples. For simplifying the cross entropy (CE) for the binary classification, $q_c$ for class $c$ is defined with the predicted probability $p_c \in [0, 1]$ of a detector for the class label $y_c = \{0, 1\}$ as: $q_c = p_c$ if $y_c = 1$. Otherwise, $q_c = 1 - p_c$. The balancing parameter $\alpha_t$ is also defined as: $\alpha_t = \alpha$ if $y_c = 1$. Otherwise, $\alpha_t = 1 - \alpha$. $\alpha \in [0, 1]$ is a weighting factor needed to be set. Then, the original focal loss based on the BCE is:

$$Focal(q_c) = -\alpha_t(1 - q_c)^\gamma \log(q_c), \tag{1}$$

$\gamma \in [0, 5]$ is a tunable focusing parameter. Due to the modulating $-\alpha_t(1 - q_c)^\gamma$, the focal loss can smoothly down-weight easy samples and focus on hard samples more and easy background sample less. However, as in Fig. 4, this loss is less sensitive to the probability of false positive responses. To address this, we present a lightness focal loss. From a detector, we can predict a bounding box $\mathbf{d} = (cx, cy, w, h)$, where $cx, cy, w$ and $h$ are center positions, width and height, and a class confidence $\mathbf{p} = [0, 1]^C$ for total $C$ light-source classes. Given an encoded target label $\mathbf{y} = \{0, 1\}^C$, we can compute $\mathbf{q} = (q_1, ..., q_C)$. Then, we define our LF loss as follows:

$$LF(\mathbf{q}, \mathbf{y}, \mathbf{d}) = -\frac{1}{C} \sum_{c=1}^{C} \alpha_t(1 - q_c)^\gamma \log(q_c) \left( y_c + (1 - y_c) \frac{\eta - \phi(\mathbf{d}_c)}{q_c + \epsilon} \right) \tag{2}$$

Compared to Eq. (1), we include the lightness spatial attention prior $\eta - \phi(\mathbf{d}_c)$ and false positive focal loss $(1/(q_c + \epsilon))$ terms. When $0.5 \leq p_c$ for $y_c = 0$, our FP focal loss increases the loss more steeply as shown in Fig. 4. Specially, it polynomially increases the loss with a factor of $(q_c + \epsilon)^{-1}$ within $0 \leq p_c \leq 1$ for $y_c = 0$ as shown in Fig. 4-(A). For example, if $p_c = 0.6$ and $p_c = 0.9$, this term increases the loss $\times 2.5$ and $\times 10$. To avoid the zero division, we add $\epsilon = 10^{-5}$.

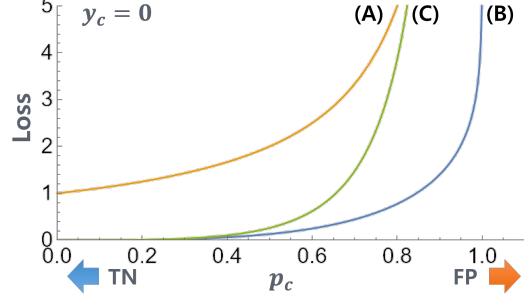

Figure 4: (A) means our FP focal loss $1/(q_c + \epsilon)$; (B) means original focal loss in Eq. (1); (C) means our LF loss in Eq. (2) with the prior $\eta - \phi(\mathbf{d}_c) = 1$. All the curves show the variation of losses when $y_c = 0$. As $p_c$ becomes 1, (A) dramatically increases the loss. Our LF loss (C) is more sensitive to the FP probability than the focal loss (B).

The spatial attention prior $\phi(\mathbf{d}, c) \in [0, 1]$ is the light occurrence probability within a region. $\eta = [2, 5]$ is a penalizing factor needed to be to tuned and we achieve the best result with $\eta = 4$ (Refer to our supplementary material). Basically, we increase the loss of a FP detection with a factor of $\eta$ linearly, but reduce its amount as the lightness prior $\phi(\mathbf{a}, c)$ becomes higher.

This lightness attention prior is pre-learned, and we provide the details in the next section. For $y_c = 1$, our LF loss is the original focal loss in Eq. (1). For $y = 0$, our additional loss terms re-weight the loss.

## 4.2 Lightness spatial attention prior

In the real world, the location of a light source attached to an object or a structure has some geometrical patterns according to each class in many cases. More specially, a streetlight is located much higher than a car or traffic lights on the road. Also, the installed position of a traffic light is higher than the one of a car light. Because a camera is usually mounted at the top front of a vehicle, we conjecture that a region of the light source in an image could have geometrical patterns.

To exploit it for LD, we then define a lightness prior map $\Phi(i, j, c) \in [0, 1]^{W \times H \times C}$ for $c$ class representing the probability of a light source occurring on a pixel $(i, j)$ in an image. For generating $\Phi(i, j, c)$, we first resize an image to a $H \times W$ resolution ($1333 \times 800$ in our experiment) and transform the location and size of a GT bounding box $\mathbf{d}_c^*$ for class $c$ according to the re-sized image. Then, we can increase the lightness score on pixels within the $\mathbf{d}_c^*$ by adding one. We then normalize the score maps with min-max normalization for each class. However, $\Phi(i, j, c)$ is a very sparse density map due to the tiny source sizes. To

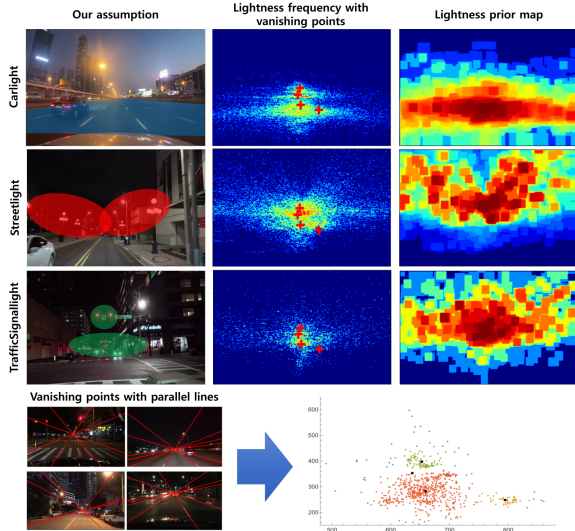

Figure 5: We illustrate the learned lightness spatial attention prior maps for different light sources. Here, the density of the maps is represented with different colors: red (high), and blue (low). We also draw our prediction for regions likely to be distributed densely of each light source in the first column. For comparison with real-world geometry, we mark vanishing points (red cross) on the maps.

remedy this, we smooth it by using max-pooling with $55 \times 55$ kernels, and then a Gaussian filter with $5 \times 5$ filter with $\sigma = 3$. We subsequently apply histogram equalization to enhance it more. Then, we can compute the lightness spatial attention prior $\phi(\mathbf{d}_c)$ with $\Phi(i, j, c)$ for a detection $\mathbf{d}_c$:
$$\phi(\mathbf{d}_c) = \frac{1}{w \times h} \sum_{i = cx - 0.5w}^{cx + 0.5w} \sum_{j = cy - 0.5h}^{cy + 0.5h} \Phi(i, j, c).$$

Figure 5 visualizes the constructed $\Phi(i, j, c)$ for different light sources on YDLD. As we assumed, we found that each light source has a distinct lightness distribution. We also draw vanishing points (VPs) and parallel lines (*e.g.* lane, building, etc) for evaluation. Because the VPs contain some global geometrical context [35] in an image, it can be used for various tasks [36–38]. In general, VPs are the nearest on the horizon, and the furthest point without any obstacles in an one-point perspective image [36]. Therefore, we consider the regions below and upper to the horizontal line through a VP as the ground and sky when a camera is fixed toward the front and a vehicle drives straight on a flat road. We depict the four centroids of accumulated VPs on training images using the DBSCAN algorithm [39].

As shown in Fig. 5, we observe that the learned lightness score map closely matches the geometrical representation by the VP. To describe it, we divide a road scene into conceptual areas according to the horizontal (*e.g.* sky and ground) and parallel lines (*e.g.* road and building lines) passing through VPs. Most Carlights are widely distributed within the ground plane nearby VPs. The Streetlights are densely located around the top-left and top-right to the VPs but are sparse below the VPs. TrafficSignallights are dense nearby the VPs, and are also sparse below the VPs. Therefore, we can embed the geometrical context of light source into a detector using the lightness spatial attention prior, and indeed improve mAP as shown in Table 8.

### 4.3 Semi-supervised focal light detection

Let denote $\mathbf{D}^l = \{\mathbf{x}_i^l, \mathbf{y}_i^l\}$ and $\mathbf{D}^u = \{\mathbf{x}_i^u\}$ as labeled and unlabeled sets, where $\mathbf{x}$ and $\mathbf{y}$ are an image and the GT labels consisting of light-source locations and classes, respectively. Basically, our SS-FLD follows the conventional SSOD training approach. So, we use weak (*i.e.* flip and resize, etc) and strong (*i.e.* rotation/flip/resize + erase) augmentations on the labeled or unlabeled images, and feed the augmented images to teacher and student detectors as shown in Fig. 3. The teacher network $f_t$ aims to generate pseudo-label $\hat{\mathbf{y}}^\mathbf{u}$ on weak-augmented unlabeled data $\mathbf{x}^u$. The student model $f_s$ is trained with both weak-augmented labeled and strong-augmented unlabeled images in which pseudo-labels is generated by the teacher detector on the same unlabeled but weak augmented image. The teacher and student networks are initialized randomly at the first iteration of training. Once the parameters $\theta_s$ of the student network are learned by minimizing a target task loss, the parameter $\theta_t$ of the teacher network can be updated using the exponential moving average (EMA) [31] with the smoothing coefficient hyperparameter $\delta$ (=0.9995 in our experiment.): $\theta_t \leftarrow \delta\theta_t + (1 - \delta)\theta_s$. Here, $\tau$ is the confidence score threshold for selecting reliable pseudo labels, and we set it as 0.5 described in [31, 32]. Finally, we define a total detection loss of our SS-FLD:

$$
\begin{aligned}
L(\mathbf{x}^l, \mathbf{y}^l, \mathbf{x}^u, \hat{\mathbf{y}}^u) \quad &= \tfrac{1}{N} \sum_{i=1}^{N} [LF(f_s(\mathbf{x}_i^l), \mathbf{y}_i^l, \mathbf{d}) + L_{reg}(f_s(\mathbf{x}_i^l), \mathbf{y}_i^l)] \\
&+ \lambda_u \tfrac{1}{M} \sum_{j=1}^{M} [LF(f_s(\mathbf{x}_j^u), \hat{\mathbf{y}}_j^u, \mathbf{d}) + L_{reg}(f_s(\mathbf{x}_j^u), \hat{\mathbf{y}}_j^u)]
\end{aligned}
\tag{3}
$$

We replace the classification loss with our LF loss Eq. 2. For the regression loss $L_{reg}$, we use GIoU loss [40]. In Table 6, we apply our SS-FLDs for the several detectors to prove the effects.

## 5 Experiments

This section provides the benchmark evaluation on YDLD and ablation study of our methods.

### 5.1 Dataset comparison

To prove the generalization ability of YDLD, we perform a transfer learning evaluation of how much the pre-training model on a certain dataset can enhance the accuracy of another dataset. To do this, we first pre-train

Table 4: Transfer learning effects of YDLD

| Dataset | $mAP$ | # Images | # GTs | # Overlapped Classes | # Countries |
|---------|-------|----------|-------|----------------------|-------------|
| ImageNet | 35.1 | 14,197,122 | N/A | 6 | N/A |
| COCO | 38.3 | 123,287 | 896,782 | 9 | N/A |
| KITTI | 35.8 | 7,481 | 51,865 | 5 | 1 |
| SODA10M | 36.2 | 10,000 | 78,239 | 5 | 1 |
| YDLD | 38.3 | 3,156 | 116,028 | 1 | 19 |

each vanilla DINO detector on YDLD, KITTI [41], SODA10M [42], ImageNet [43] and MS-COCO [26]. Then, we fine-tune and evaluate each DINO detector on BDD100k [44]. Three driving datasets (KITTI, SODA10M, and BDD100k) share the common five classes (pedestrian, car, truck, tram, and cyclist), and only BDD100k has the traffic light class. For a fair comparison, all training is conducted

for 12 epochs with the same hyper-parameters as the DINO original setting. Table 4 provides overall summary. We provide the number of the overlapped classes with the classes BDD100k and target countries.

Remarkably, the pre-trained DINO on our YDLD shows the best mAP gains even though the overlapped classes of YDLD to BDD100k are the fewest among the datasets. We expect this improvement is due to the diversity of the image scenes and target countries with different road environments as shown in Table 2 and 4. Also, our dataset has a similar scene structure since both datasets contain driving scene images. These results show the sufficient diversity and generality of our YDLD. In addition, our YDLD can be complementary to other driving datasets.

## 5.2 Implementation details

For fair benchmark evaluation, we train all detectors with a ResNet-50 backbone [45] on YDLD dataset. For implementing our SS-FLDs, we adopt the representatives of tiny, transformer-based, and single-stage detectors: RFLA [7], DINO [28], and PAA [27] as baseline detectors. We implement RFLA w/t PAA by replacing the IoU-based label assignment with the Gaussian receptive field-based assignment. Therefore, RFLA first assigns GTs and anchors. Then, the probabilistic anchor assignment is employed to reassign them.

For semi-supervision, we build extra unlabeled 18.5k images collected from a vehicle black box and YouTube under urban driving scenarios. To stabilize the training, we use a warm-up phase on the unlabeled data for the initial 10k iterations [32]. We train a detector without/with SSOD for 12 epochs and 40k iterations. But, Deformable DETR and YOLOX are trained for 50 and 300 epochs, respectively. We set $\alpha$ and $\gamma$ to 0.25 and 2, respectively. We optimize the detector with the SGD method with a learning rate of 0.01, a momentum of 0.9, and a weight decay of 0.001. We implement all the detectors using the MMDetection [46] with NVIDIA RTX A6000 GPUs.

## 5.3 YDLD benchmark evaluation

In Table 5, we conduct the benchmark evaluation over different types of 20 detectors The classical one-stage and two-stage detectors such as Faster R-CNN, YOLOF, and RetinaNet show low mAP scores. On the other hand, the DINO and Deformable DETR of the transformer variants show better mAP scores. It shows global attention feature is beneficial for LD since relational contexts between light sources and nearby objects can be encoded. In a similar context, we design the lightness spatial attention prior.

Since a small localization error between boxes degrades an IoU score rapidly [10], the improved label assignment methods are useful for detecting smaller objects. Consequently, PAA, ATSS, and YOLOX with the improved label assignment show higher scores than other classical detectors. Also, the improved assignment is effective for tiny object detectors when comparing RFLA w/t RetinaNet and RFLA w/t PAA. Compared to the recent detectors, our SS-FLDs achieve the best 26.0 mAP score and outperform other detectors for all the metrics. Especially, they boost the $AP_{vt}$, $AP_t$, and $AP_s$ scores greatly which are critical metrics of TOD.

Table 5: YDLD benchmark evaluation on the several types of detectors. We highlight the best record with **red bold**. † represents our implementation.

| Detectors | $mAP$ | $AP_{vt}$ | $AP_t$ | $AP_s$ |
|---|---|---|---|---|
| **Two-stage object detectors** | | | | |
| Libra R-CNN [47] | 8.6 | 0.2 | 6.1 | 10.2 |
| Faster R-CNN [48] | 8.9 | 0.1 | 6.2 | 10.5 |
| Cascade R-CNN [49] | 10.2 | 0.3 | 7.3 | 12.5 |
| **Single-stage object detectors** | | | | |
| YOLOF [50] | 5.7 | 0.0 | 3.7 | 7.2 |
| RetinaNet [34] | 8.1 | 2.2 | 4.8 | 20.6 |
| ATSS [51] | 15.6 | 4.7 | 11.7 | 25.3 |
| YOLOX [52] | 17.9 | 8.1 | 14.4 | 27.5 |
| PAA [27] | 21.6 | 9.7 | 18.0 | 34.6 |
| **Transformer-based object detectors** | | | | |
| Deformable DETR [5] | 16.7 | 7.3 | 14.0 | 26.4 |
| DINO [28] | 22.6 | 10.5 | 19.0 | 35.2 |
| **Semi-supervised object detectors** | | | | |
| SoftTeacher [53] | 8.4 | 0.3 | 6.4 | 9.3 |
| MeanTeacher [31, 32] | 14.4 | 9.0 | 16.0 | 25.2 |
| ConsistentTeacher [32] | 19.1 | 7.4 | 15.0 | 31.7 |
| **Tiny object detectors** | | | | |
| CEASC [54] | 7.5 | 2.3 | 4.7 | 18.5 |
| FSANet [25] | 10.4 | 2.3 | 7.7 | 16.1 |
| RFLA w/t RetinaNet [7] | 10.8 | 5.9 | 9.8 | 16.9 |
| NWD-RKA [55] | 16.6 | 7.3 | 14.0 | 26.4 |
| RFLA w/t PAA † | 21.6 | 9.7 | 18.0 | 34.6 |
| SS-FLD w/t DINO | 25.6 | 12.7 | 23.1 | 38.6 |
| SS-FLD w/t RFLA | **26.0** | **12.8** | **24.8** | **39.2** |

These results show that LD is very challenging due to intrinsic characteristics of similar appearance and tiny sizes. As a result, many false detections can be generated as shown in Fig. 7. Exploiting the improved label assignment, global attention with a transformer, and SSOD methods are also

beneficial for handling the challenges. However, we confirm that our methods are more appropriate for LD by suppressing false positives and learning global structures.

## 5.4 Effects of SS-FLD

To demonstrate effects of our SS-FLD, we conduct the comparisons with SS-FLDs and various baseline detectors which use the focal loss. Then, we replace the focal loss with our LF loss, and apply our SS-FLD method for each detector. To apply SS-FLD methods for these baselines, we exploit the data augmentation and EMA as mentioned in Sec. 4.3. Note that, others (*e.g.* network architecture and label assignment) are not altered. We train them for the same 40k iterations.

Table 6: Effects of our semi-supervised focal light detection (SS-FLD) on the YDLD.

| Baseline | SS-FLD | $mAP$ | $AP_{vt}$ | $AP_t$ | $AP_s$ |
|---|---|---|---|---|---|
| RFLA w/t RetinaNet | | 10.8 | 5.9 | 9.8 | 16.9 |
| RFLA w/t RetinaNet | ✓ | 18.6 [↑ 7.8] | 8.5 [↑ 2.6] | 17.0 [↑ 7.2] | 29.4 [↑ 12.5] |
| RFLA w/t PAA | | 21.6 | 11.0 | 19.4 | 33.4 |
| RFLA w/t PAA | ✓ | 26.0 [↑ 4.4] | 12.8 [↑ 1.8] | 24.8 [↑ 5.4] | 39.2 [↑ 5.8] |
| ATSS | | 15.6 | 4.7 | 11.7 | 25.3 |
| ATSS | ✓ | 17.7 [↑ 2.1] | 6.0 [↑ 1.3] | 13.3 [↑ 1.6] | 29.2 [↑ 3.9] |
| PAA | | 21.6 | 9.7 | 18.0 | 34.6 |
| PAA | ✓ | 25.6 [↑ 4.0] | 11.9 [↑ 2.2] | 22.4 [↑ 4.4] | 40.1 [↑ 5.5] |
| Deformable DETR | | 16.7 | 7.3 | 14.0 | 26.4 |
| Deformable DETR | ✓ | 18.0 [↑ 1.3] | 8.7 [↑ 1.4] | 15.8 [↑ 1.8] | 28.4 [↑ 2.0] |
| DINO | | 22.6 | 10.5 | 19.0 | 35.2 |
| DINO | ✓ | 25.6 [↑ 3.0] | 12.7 [↑ 2.2] | 23.1 [↑ 4.1] | 38.6 [↑ 3.4] |

As shown in Table 6, our SS-FLD considerably increases mAP scores about 7.8, 4.0, and 3.0 points for RFLA, PAA, and DINO, respectively. Compared with other baseline detectors, our SS-FLD method demonstrates remarkable improvements. In Fig 6, we provide the more qualitative comparison of learned attention maps.

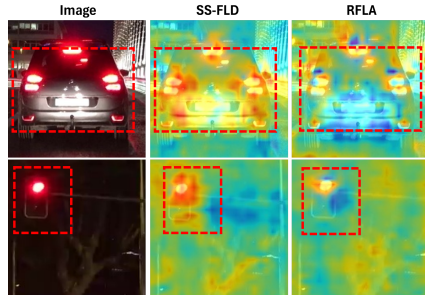

## 5.5 Ablation experiments

In this section, we prove effects of the LF loss and SSOD methods. We investigate the mAP gains by adding each method one-by-one to the baseline detectors: RFLA and PAA. We use the same implementation manner as in Sec. 5.4.

Figure 6: We visualize attention maps of RFLA w/t and w/o SS-FLD using Grad-CAM [56] (red indicates higher, but blue lower scores). We highlight a region with distinct attention differences with red dotted boxes.

As shown in Table 7, our LF loss increases mAP scores by about 6.7 and 2.9 for RFLA and PAA, respectively. This result shows the effects of the lightness attention prior and the FP focal loss. More mAP gains can be achieved by using our semi-supervision by 0.7 and 1.1 for each detector.

## 5.6 Detailed analysis of LF loss

Table 8 demonstrates the effectiveness of each term of LF loss. We implement different versions of SS-FLDs by changing the focal loss as follows: (M1) Original focal loss, (M2) LF loss with the FP focal loss only, (M3) LF loss with the lightness spatial attention prior only, (M4) LF loss with both terms. When comparing (M1) with (M2) and (M3), the LF loss and attention prior greatly improve mAPs by 2.8 and 2.9. When using both, we can further improve mAP by 4.0.

Table 7: Effects of the LF loss and SSOD method on the YDLD dataset.

| Baseline | LF | SSOD | $mAP$ | $AP_{50}$ | $AP_{75}$ | $AP_{vt}$ | $AP_t$ |
|---|---|---|---|---|---|---|---|
| RFLA | | | 21.6 | 50.8 | 14.7 | 11.0 | 19.4 |
| RFLA | ✓ | | 24.3 | 54.5 | 17.7 | 11.9 | 22.2 |
| RFLA | ✓ | ✓ | 26.0 | 58.3 | 19.0 | 12.8 | 24.8 |
| PAA | | | 21.6 | 48.0 | 16.1 | 9.7 | 18.0 |
| PAA | ✓ | | 24.5 | 52.1 | 19.8 | 11.1 | 21.0 |
| PAA | ✓ | ✓ | 25.6 | 55.1 | 20.4 | 11.9 | 22.4 |

## 5.7 Qualitative result

Figure 7 shows the detection result comparisons on YDLD. Our SS-FLD shows better detection results. In some cases, other detectors produce false negative (cyan circle) and false positive (magenta circle) detections. For the reflected and glare lights, ConsistentTeacher, PAA, and DINO predict inaccurate detection results. But, our detector successfully excludes them. This comparison results show the superiority of our SS-FLD again.

Table 8: Comparison of different LF losses.

| Methods | $mAP$ | $AP_{50}$ | $AP_{75}$ | $AP_{vt}$ | $AP_t$ |
|---|---|---|---|---|---|
| M1 | 21.6 | 48.0 | 16.1 | 9.7 | 18.0 |
| M2 | 24.4 | 51.8 | 19.6 | 11.4 | 20.9 |
| M3 | 24.5 | 52.0 | 19.7 | 11.3 | 21.3 |
| M4 | 25.6 | 55.1 | 20.4 | 11.9 | 22.4 |

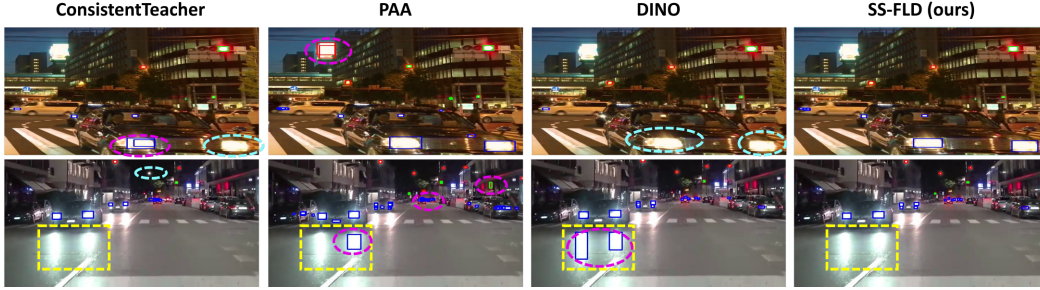

Figure 7: Detection results using our SS-FLD, ConsistentTeacher, PAA, and DINO are compared. Red, blue, and green boxes represent Streetlight, Carlight, and TrafficSignallight classes, respectively. Reflected lights are marked with yellow boxes.

# 6 Conclusion

We launch a new YDLD benchmark challenge for multi-source light detection due to the less attention to this problem. We provide many details of the dataset construction and statistics. In particular, we prove the quality and scale of our dataset from the comparison of other solid benchmark datasets using transfer learning evaluation. From the evaluation of different sorts of 20 detectors on YDLD, we compare their detection results and analyze the pros and cons of each method.

To improve LD further, we present a new LF loss and improved SSOD. The main effects of the LF loss suppress false positives by embedding the lightness spatial attention prior. From the comparison with the state-of-the-art detectors, we prove that our SS-FLD outperforms them. By applying our methods to other detectors, we can boost their accuracy greatly and consistently. We hope that our work is a solid baseline for the multi-source LD. Yet, YDLD has not yet included the sunlight class because the annotation of this class is very costly due to the non-regular shapes and frequent glares. In our future work, we will provide pixel-wise labeling including sunlight, and expect that the YDLD can be extended to instance segmentation or dense prediction challenges.

**The future direction of YDLD** Light detection can be a fundamental and core technology in handling other downstream tasks. Most of all, LD is beneficial for understanding the behaviors of vehicles in night driving. Therefore, we will plan to launch a vehicle path and action recognition benchmark challenge based on vehicle action recognition methods [57–59]. To this end, we will expand the class taxonomy of the YDLD dataset (e.g., vehicle headlights, taillights, turn signals, etc). In addition, LD is also very essential for glare or artifact removals [19, 18, 20] since discrimination between the light source and other sources is needed. Therefore, the extended benchmark of light detection and glare removal is planned in our next research. Furthermore, de-blurring for low light images [60] can be another crucial extension using YDLD since inter camera motion between consecutive frames can be predicted by detecting light sources [61, 62].

# 7 Limitations

YDLD does not consider indoor lights or signboards as foreground until now since they are less significant for the other three classes under night driving. However, to improve the generality of YDLD, we will treat the neon signs, signboards, indoor lighting, and decorative lights as distinct light-source classes. In addition, as mentioned Sec. 3.1, light source detection is very effective in understanding other object contexts. Therefore, it is very worth including object annotations related to lights such as a vehicle or pole to extend this work to other tasks.

## Acknowledgments and Disclosure of Funding

This work was supported in part by the National Research Foundation of Korea (NRF) grants funded by the Korea government (MSIT) (No. NRF-2022R1C1C1009208) and funded by the Ministry of Education (No.2022R1A6A1A03051705); supported in part by Institute of Information & communications Technology Planning & Evaluation (IITP) grants funded by the Korea government (MSIT) (No.2022-0-00448/RS-2022-II220448: Deep Total Recall, 30%, No.RS-2022-00155915: Artificial Intelligence Convergence Innovation Human Resources Development (Inha University)); supported in part by INHA UNIVERSITY Research Grant.

## References

[1] L. Yahiaoui, M. Uřičář, A. Das, and S. Yogamani, "Let the sunshine in: Sun glare detection on automotive surround-view cameras," *Electronic Imaging*, vol. 2020, no. 16, pp. 80–1, 2020.

[2] Y. Chen, F. Liu, and K. Pei, "Self-supervised sun glare detection cnn for self-aware autonomous driving," in *NeurIPS Workshop*, 2021.

[3] K. He, G. Gkioxari, P. Dollár, and R. Girshick, "Mask r-cnn," in *ICCV*, pp. 2961–2969, 2017.

[4] N. Carion, F. Massa, G. Synnaeve, N. Usunier, A. Kirillov, and S. Zagoruyko, "End-to-end object detection with transformers," in *ECCV*, vol. 12346, pp. 213–229.

[5] X. Zhu, W. Su, L. Lu, B. Li, X. Wang, and J. Dai, "Deformable detr: Deformable transformers for end-to-end object detection," in *ICLR*, 2021.

[6] S. H. Bae, "Deformable part region learning and feature aggregation tree representation for object detection," *TPAMI*, vol. 45, no. 9, pp. 10817–10834, 2023.

[7] C. Xu, J. Wang, W. Yang, H. Yu, L. Yu, and G.-S. Xia, "Rfla: Gaussian receptive field based label assignment for tiny object detection," in *ECCV*, pp. 526–543, 2022.

[8] C. Yang, Z. Huang, and N. Wang, "Querydet: Cascaded sparse query for accelerating high-resolution small object detection," in *CVPR*, pp. 13668–13677, 2022.

[9] X. Yu, Y. Gong, N. Jiang, Q. Ye, and Z. Han, "Scale match for tiny person detection," in *IEEE Winter Conference on Applications of Computer Vision*, pp. 1257–1265, 2020.

[10] G. Cheng, X. Yuan, X. Yao, K. Yan, Q. Zeng, and J. Han, "Towards large-scale small object detection: Survey and benchmarks," *arXiv*, 2022.

[11] S. Pharadornpanitchakul and R. Chaisricharoen, "Danger detection from head light glare through vision estimation," in *International Symposium on Communications and Information Technologies*, pp. 661–664, 2013.

[12] M. Andalibi and D. M. Chandler, "Automatic glare detection via photometric, geometric, and global positioning information," *Electronic Imaging*, vol. 29, pp. 77–82, 2017.

[13] M. Singh, R. K. Tiwari, K. Swami, and A. Vijayvargiya, "Detection of glare in night photography," in *International Conference on Pattern Recognition*, pp. 865–870, 2016.

[14] J. Bernse, "Dynamic thresholding of grey-level images," in *International Conference on Pattern Recognition*, pp. 1251–1255, 1986.

[15] N. Otsu, "A threshold selection method from gray-level histograms," *IEEE Transactions on Systems, Man, and Cybernetics*, vol. 9, no. 1, pp. 62–66, 1979.

[16] D. Rodin and N. Orlov, "Fast glare detection in document images," in *International Conference on Document Analysis and Recognition Workshops*, vol. 7, pp. 6–9, 2019.

[17] J.-Y. Zhu, T. Park, P. Isola, and A. A. Efros, "Unpaired image-to-image translation using cycle-consistent adversarial networks," in *ICCV*, pp. 2223–2232, 2017.

[18] X. Qiao, G. P. Hancke, and R. W. Lau, "Light source guided single-image flare removal from unpaired data," in *ICCV*, pp. 4177–4185, 2021.

[19] Y. Wu, Q. He, T. Xue, R. Garg, J. Chen, A. Veeraraghavan, and J. T. Barron, "How to train neural networks for flare removal," in *ICCV*, pp. 2239–2247, October 2021.

[20] Y. Dai, C. Li, S. Zhou, R. Feng, and C. C. Loy, "Flare7k: A phenomenological nighttime flare removal dataset," in *NeurIPS*, vol. 35, pp. 3926–3937, 2022.

[21] A. Sharma and R. T. Tan, "Nighttime visibility enhancement by increasing the dynamic range and suppression of light effects," in *CVPR*, pp. 11972–11981, 2021.

[22] Y. Jin, W. Yang, and R. T. Tan, "Unsupervised night image enhancement: When layer decomposition meets light-effects suppression," in *ECCV*, pp. 404–421, Springer, 2022.

[23] G. Xia, X. Bai, J. Ding, Z. Zhu, S. J. Belongie, J. Luo, M. Datcu, M. Pelillo, and L. Zhang, "DOTA: A large-scale dataset for object detection in aerial images," in *CVPR*, pp. 3974–3983, 2018.

[24] J. Wang, W. Yang, H. Guo, R. Zhang, and G.-S. Xia, "Tiny object detection in aerial images," in *International Conference on Pattern Recognition*, pp. 3791–3798, 2021.

[25] J. Wu, Z. Pan, B. Lei, and Y. Hu, "Fsanet: Feature-and-spatial-aligned network for tiny object detection in remote sensing images," *IEEE Transactions on Geoscience and Remote Sensing*, vol. 60, pp. 1–17, 2022.

[26] T.-Y. Lin, M. Maire, S. Belongie, J. Hays, P. Perona, D. Ramanan, P. Dollár, and C. L. Zitnick, "Microsoft coco: Common objects in context," in *ECCV*, pp. 740–755, 2014.

[27] K. Kim and H. S. Lee, "Probabilistic anchor assignment with iou prediction for object detection," in *ECCV*, pp. 355–371, 2020.

[28] H. Zhang, F. Li, S. Liu, L. Zhang, H. Su, J. Zhu, L. M. Ni, and H. Shum, "DINO: DETR with improved denoising anchor boxes for end-to-end object detection," in *ICLR*, 2023.

[29] M. A. Esfahani and H. Wang, "Robust glare detection: Review, analysis, and dataset release," *arXiv*, 2021.

[30] K. Yoneda, N. Ichihara, H. Kawanishi, T. Okuno, L. Cao, and N. Suganuma, "Sun-glare region recognition using visual explanations for traffic light detection," in *Intelligent Vehicles Symposium*, pp. 1464–1469, IEEE, 2021.

[31] A. Tarvainen and H. Valpola, "Mean teachers are better role models: Weight-averaged consistency targets improve semi-supervised deep learning results," in *NeurIPS*, pp. 1195–1204, 2017.

[32] X. Wang, X. Yang, S. Zhang, Y. Li, L. Feng, S. Fang, C. Lyu, K. Chen, and W. Zhang, "Consistent-teacher: Towards reducing inconsistent pseudo-targets in semi-supervised object detection," in *CVPR*, pp. 3240–3249, 2023.

[33] Y. Liu, C. Ma, Z. He, C. Kuo, K. Chen, P. Zhang, B. Wu, Z. Kira, and P. Vajda, "Unbiased teacher for semi-supervised object detection," in *ICLR*, 2021.

[34] T.-Y. Lin, P. Goyal, R. Girshick, K. He, and P. Dollár, "Focal loss for dense object detection," in *ICCV*, pp. 2980–2988, 2017.

[35] R. T. Collins and R. S. Weiss, "Vanishing point calculation as a statistical inference on the unit sphere," in *ICCV*, pp. 400–403, 1990.

[36] S. Lee, J. Kim, J. S. Yoon, S. Shin, O. Bailo, N. Kim, T. Lee, H. S. Hong, S. Han, and I. S. Kweon, "Vpgnet: Vanishing point guided network for lane and road marking detection and recognition," in *ICCV*, pp. 1965–1973, 2017.

[37] Q. Wang, T. Han, Z. Qin, J. Gao, and X. Li, "Multitask attention network for lane detection and fitting," *IEEE Transactions on Neural Networks and Learning Systems*, vol. 33, no. 3, pp. 1066–1078, 2022.

[38] H. Kong, J. Audibert, and J. Ponce, "Vanishing point detection for road detection," in *CVPR*, pp. 96–103, 2009.

[39] M. Ester, H.-P. Kriegel, J. Sander, X. Xu, *et al.*, "A density-based algorithm for discovering clusters in large spatial databases with noise," in *Proceedings of the Second International Conference on Knowledge Discovery and Data Mining*, vol. 96, pp. 226–231, 1996.

[40] H. Rezatofighi, N. Tsoi, J. Gwak, A. Sadeghian, I. Reid, and S. Savarese, "Generalized intersection over union: A metric and a loss for bounding box regression," in *CVPR*, pp. 658–666, 2019.

[41] A. Geiger, P. Lenz, and R. Urtasun, "Are we ready for autonomous driving? the KITTI vision benchmark suite," in *CVPR*, pp. 3354–3361, 2012.

[42] J. Han, X. Liang, H. Xu, K. Chen, L. Hong, J. Mao, C. Ye, W. Zhang, Z. Li, X. Liang, and C. Xu, "SODA10M: A large-scale 2d self/semi-supervised object detection dataset for autonomous driving," in *NeurIPS*, 2021.

[43] J. Deng, W. Dong, R. Socher, L. Li, K. Li, and L. Fei-Fei, "Imagenet: A large-scale hierarchical image database," in *CVPR*, pp. 248–255, 2009.

[44] F. Yu, H. Chen, X. Wang, W. Xian, Y. Chen, F. Liu, V. Madhavan, and T. Darrell, "BDD100K: A diverse driving dataset for heterogeneous multitask learning," in *CVPR*, pp. 2633–2642, 2020.

[45] K. He, X. Zhang, S. Ren, and J. Sun, "Deep residual learning for image recognition," in *CVPR*, pp. 770–778, 2016.

[46] K. Chen, J. Wang, J. Pang, Y. Cao, Y. Xiong, X. Li, S. Sun, W. Feng, Z. Liu, J. Xu, Z. Zhang, D. Cheng, C. Zhu, T. Cheng, Q. Zhao, B. Li, X. Lu, R. Zhu, Y. Wu, J. Dai, J. Wang, J. Shi, W. Ouyang, C. C. Loy, and D. Lin, "MMDetection: Open mmlab detection toolbox and benchmark," *arXiv*, 2019.

[47] J. Pang, K. Chen, J. Shi, H. Feng, W. Ouyang, and D. Lin, "Libra r-cnn: Towards balanced learning for object detection," in *CVPR*, pp. 821–830, 2019.

[48] S. Ren, K. He, R. Girshick, and J. Sun, "Faster r-cnn: Towards real-time object detection with region proposal networks," in *NeurIPS*, vol. 28, 2015.

[49] Z. Cai and N. Vasconcelos, "Cascade r-cnn: Delving into high quality object detection," in *CVPR*, pp. 6154–6162, 2018.

[50] Q. Chen, Y. Wang, T. Yang, X. Zhang, J. Cheng, and J. Sun, "You only look one-level feature," in *CVPR*, pp. 13039–13048, 2021.

[51] S. Zhang, C. Chi, Y. Yao, Z. Lei, and S. Z. Li, "Bridging the gap between anchor-based and anchor-free detection via adaptive training sample selection," in *CVPR*, pp. 9759–9768, 2020.

[52] Z. Ge, S. Liu, F. Wang, Z. Li, and J. Sun, "Yolox: Exceeding yolo series in 2021," *arXiv*, 2021.

[53] M. Xu, Z. Zhang, H. Hu, J. Wang, L. Wang, F. Wei, X. Bai, and Z. Liu, "End-to-end semi-supervised object detection with soft teacher," *ICCV*, 2021.

[54] B. Du, Y. Huang, J. Chen, and D. Huang, "Adaptive sparse convolutional networks with global context enhancement for faster object detection on drone images," in *CVPR*, pp. 13435–13444, 2023.

[55] C. Xu, J. Wang, W. Yang, H. Yu, L. Yu, and G.-S. Xia, "Detecting tiny objects in aerial images: A normalized wasserstein distance and a new benchmark," *ISPRS Journal of Photogrammetry and Remote Sensing*, vol. 190, pp. 79–93, 2022.

[56] R. R. Selvaraju, M. Cogswell, A. Das, R. Vedantam, D. Parikh, and D. Batra, "Grad-cam: Visual explanations from deep networks via gradient-based localization," in *ICCV*, pp. 618–626, 2017.

[57] Y. Ma, X. Zhu, S. Zhang, R. Yang, W. Wang, and D. Manocha, "Trafficpredict: Trajectory prediction for heterogeneous traffic-agents," in *AAAI*, vol. 33, pp. 6120–6127, 2019.

[58] S. Malla, B. Dariush, and C. Choi, "Titan: Future forecast using action priors," in *CVPR*, pp. 11186–11196, 2020.

[59] S. Ettinger, S. Cheng, B. Caine, C. Liu, H. Zhao, S. Pradhan, Y. Chai, B. Sapp, C. R. Qi, Y. Zhou, *et al.*, "Large scale interactive motion forecasting for autonomous driving: The waymo open motion dataset," in *ICCV*, pp. 9710–9719, 2021.

[60] C. Zhou, M. Teng, J. Han, J. Liang, C. Xu, G. Cao, and B. Shi, "Deblurring low-light images with events," *International Journal of Computer Vision*, vol. 131, no. 5, pp. 1284–1298, 2023.

[61] S. Nah, S. Son, and K. M. Lee, "Recurrent neural networks with intra-frame iterations for video deblurring," in *CVPR*, pp. 8102–8111, 2019.

[62] H. Zhang and J. Yang, "Intra-frame deblurring by leveraging inter-frame camera motion," in *CVPR*, pp. 4036–4044, 2015.

